# Matrix Completion for Multi-label Image Classification

**Ricardo S. Cabral**[†,‡] **Fernando De la Torre**[‡]
[‡]Carnegie Mellon University,
Pittsburgh, PA

**João P. Costeira**[†]**, Alexandre Bernardino**[†]
[†]ISR - Instituto Superior Técnico,
Lisboa, Portugal

rscabral@cmu.edu, ftorre@cs.cmu.edu, {jpc,alex}@isr.ist.utl.pt

## Abstract

Recently, image categorization has been an active research topic due to the urgent need to retrieve and browse digital images via semantic keywords. This paper formulates image categorization as a multi-label classification problem using recent advances in matrix completion. Under this setting, classification of testing data is posed as a problem of completing unknown label entries on a data matrix that concatenates training and testing features with training labels. We propose two convex algorithms for matrix completion based on a Rank Minimization criterion specifically tailored to visual data, and prove its convergence properties. A major advantage of our approach w.r.t. standard discriminative classification methods for image categorization is its robustness to outliers, background noise and partial occlusions both in the feature and label space. Experimental validation on several datasets shows how our method outperforms state-of-the-art algorithms, while effectively capturing semantic concepts of classes.

## 1 Introduction

With the ever-growing amount of digital image data in multimedia databases, there is a great need for algorithms that can provide effective semantic indexing. Categorizing digital images using keywords, however, is the quintessential example of a challenging classification problem. Several aspects contribute to the difficulty of the image categorization problem, including the large variability in appearance, illumination and pose of different objects. Moreover, in the multi-label setting the interaction between objects also needs to be modeled.

Over the last decade, progress in the image classification problem has been achieved by using more powerful classifiers and building or learning better image representations. On one hand, standard discriminative approaches such as Support Vector Machines or Boosting have been extended to the multi-label case [28, 14] and incorporated under frameworks such as Multiple Instance Learning [31, 33, 32, 20, 27] and Multi-task Learning [26]. However, a major limitation of discriminative approaches is the lack of robustness to outliers and missing data. Recall most discriminative approaches project the data directly onto linear or non-linear spaces, thus lacking a noise model for it. To address this issue, we propose formulating the image classification problem under a matrix completion framework, that has been fueled by recent advances in Rank Minimization [7, 18]. Using this paradigm, we can easily deal with incomplete descriptions and errors in features and labels. On the other hand, traversal to the use of more powerful classifiers, better image representations, such as SIFT [17] or GIST [21] have boosted recognition and categorization performance. A common approach to represent an object has been to group local descriptors using the bag of words model [24]. Our algorithms make use of the fact that in this model the histogram of an entire image contains information of all of its subparts. By modeling the error in the histogram, our matrix completion algorithm is able to capture semantically discriminative portions of the image, thus obviating the need for training with precise localization, as required by previous methods [31, 33, 32, 20, 27].

Our main contributions are twofold: (1) We propose two new Rank Minimization algorithms, MC-Pos and MC-Simplex, motivated by the image categorization problem. We study the advantages of matrix completion over classic discriminative approaches and show that performing classification under this paradigm not only improves state-of-the-art results on several datasets, but it does so without recurring to bounding boxes or other precise localization methods in its labeling or modeling. (2) We prove that MC-Pos and MC-Simplex enjoy the same convergence properties of Fixed Point Continuation methods for Rank Minimization without constraints. We also show that this result extends to the framework presented by [11], whose convergence was only empirically verified.

## 2 Previous Work

This section reviews related work in the area of image categorization and the problem of Matrix Completion using a Rank Minimization criterion, optimized with Nuclear Norm methods.

**Image Categorization**    Since the seminal work of Barnard *et al*. [3], many researchers have addressed the problem of associating words to images. Image semantic understanding is now typically formulated as a multi-label problem. In this setting, each image may be simultaneously categorized into more than one of a set of predefined categories. An important difference between multi-class classification and multi-label classification is that classes in multi-class classification are assumed to be mutually exclusive whereas in multi-label classification are normally interdependent from one another. Therefore, many multi-class techniques such as SVM, LDA and Boosting have been modified to make use of label correlations to improve multi-label classification performance [28, 14].

Additionally, Multiple Instance Learning (MIL) approaches can be used to explicitly model the relations between labels and specific regions of the image, as initially proposed by Maron *et al*. [19]. This framework allows for the localization and classification tasks to benefit from each other, thus reducing noise in the corresponding feature space and making the learned semantic models more accurate [31, 33, 32, 20, 27, 26]. Although promising, the MIL framework is combinatorial, so several approaches have been proposed to avoid local minima and deal with the prohibitive number of possible subregions in an image. Zha *et al*. [32] make use of hidden CRFs while Vijayanarasimhan *et al*. [27] recur to multi-set kernels to emphasize instances differently. Yang *et al*. [31] exploit asymmetric loss functions to balance false positives and negatives. These methods, however, require an explicit enumeration of instances in the image. This is usually obtained by pre-segmenting images to a small fixed number of parts or applied in settings where detectors perform well, such as the problem of associating faces to captioned names [4]. On the other hand, to avoid explicitly enumerating the instances, Nguyen *et al*. [20] couple constraint generation algorithms with a branch and bound method for fast localization. Multi-task learning has also been proposed as a way to regularize the MIL problem, so as to avoid local minima due to many available degrees of freedom. In this setting, the MIL problem is jointly learned with an easier fully supervised task such as geometric context [26].

**Matrix Completion using Rank Minimization**    Rank Minimization has recently received much attention due to its success in matrix completion problems such as the Netflix challenge, where one wishes to predict a user's movie preferences based on a subset of his and other people's choices, or minimum order control [10], where the goal is to find the least complex controller achieving some performance measure.

A major breakthrough by [7] states the minimization of the rank function, under broad conditions, can be achieved using the minimizer obtained with the Nuclear Norm (sum of singular values). Since the natural reformulation of the Nuclear Norm gives rise to a Semidefinite Program, existing interior point methods can only handle problems with a number of variables in the order of the hundreds. Thus, several methods have been devised to perform this optimization efficiently [15, 6, 18, 25, 13, 1, 7, 2]. In the last few years, incremental matrix completion methods have also been proposed [1, 2, 5].

In the context of Computer Vision, minimization of the Nuclear Norm has been applied to several problems: Structure from Motion [1, 8, 5], Robust PCA [29], Subspace Alignment [22], Subspace Segmentation [16] and Tag Refinement [34].

# 3 Multi-label classification using Matrix Completion

In a supervised setting, a classifier learns a mapping[1] $\mathcal{W} : \mathcal{X} \to \mathcal{Y}$ between the space of features $\mathcal{X}$ and the space of labels $\mathcal{Y}$, from $N_{tr}$ tuples of known features and labels. Linear classifiers define $(\mathbf{x}_j, \mathbf{y}_j) \in \mathbb{R}^F \times \mathbb{R}^K$, where $F$ is the feature dimension and $K$ the number of classes, and minimize the loss $l$ between the output space and the projection of the input space, as

$$\underset{\mathbf{W}, \mathbf{b}}{\text{minimize}} \sum_{j=1}^{N_{tr}} l \left( \mathbf{y}_j, [\mathbf{W} \ \mathbf{b}] \left[ \begin{array}{c} \mathbf{x}_j \\ 1 \end{array} \right] \right), \tag{1}$$

with parameters $\mathbf{W} \in \mathbb{R}^{K \times F}$, $\mathbf{b} \in \mathbb{R}^K$. Given (1), Goldberg *et al.* [11] note that the problem of classifying $N_{tst}$ test entries can be cast as a Matrix Completion. For this purpose, they concatenate all labels and features into matrices $\mathbf{Y}_{tst} \in \mathbb{R}^{K \times N_{tst}}, \mathbf{Y}_{tr} \in \mathbb{R}^{K \times N_{tr}}, \mathbf{X}_{tst} \in \mathbb{R}^{F \times N_{tst}}, \mathbf{X}_{tr} \in \mathbb{R}^{F \times N_{tr}}$. If the linear model holds, then the matrix

$$\mathbf{Z}_0 = \left[ \begin{array}{cc} \mathbf{Y}_{tr} & \mathbf{Y}_{tst} \\ \mathbf{X}_{tr} & \mathbf{X}_{tst} \\ \mathbf{1}^\top \end{array} \right], \tag{2}$$

should be rank deficient. The classification process consists in filling the unknown entries in $\mathbf{Y}_{tst}$ such that the Nuclear Norm of $\mathbf{Z}_0$, the convex envelope of rank [7], is minimized. Since in practice we may have errors and partial knowledge in the training labels and in the feature space, let us define $\Omega_X$ and $\Omega_Y$ as the set of known feature and label entries and zero out unknown entries in $\mathbf{Z}_0$. Additionally, let the data matrix $\mathbf{Z}$ be defined as a sum of $\mathbf{Z}_0$ with an error term $\mathbf{E}$, as

$$\mathbf{Z} = \left[ \begin{array}{c} \mathbf{Z}_Y \\ \mathbf{Z}_X \\ \mathbf{Z}_1 \end{array} \right] = \left[ \begin{array}{cc} \mathbf{Y}_{tr} & \mathbf{Y}_{tst} \\ \mathbf{X}_{tr} & \mathbf{X}_{tst} \\ \mathbf{1}^\top \end{array} \right] + \left[ \begin{array}{cc} \mathbf{E}_{\mathbf{Y}tr} & \mathbf{0} \\ \mathbf{E}_{\mathbf{X}tr} & \mathbf{E}_{\mathbf{X}tst} \\ \mathbf{0}^\top \end{array} \right] = \mathbf{Z}_0 + \mathbf{E}, \tag{3}$$

where $\mathbf{Z}_Y, \mathbf{Z}_X, \mathbf{Z}_1$ respectively stand for the label, feature and last rows of $\mathbf{Z}$. Then, classification can be posed as an optimization problem that finds the best label assignment $\mathbf{Y}_{tst}$ and error matrix $\mathbf{E}$ such that the rank of $\mathbf{Z}$ is minimized. The resulting optimization problem, MC-1 [11], is

$$\underset{\mathbf{Y}_{tst}, \mathbf{E}_{\mathbf{X}tr}, \mathbf{E}_{\mathbf{Y}tr}, \mathbf{E}_{\mathbf{X}tst}}{\text{minimize}} \quad \mu \|\mathbf{Z}\|_* + \frac{1}{|\Omega_X|} \sum_{ij \in \Omega_X} c_x(z_{ij}, z_{0ij}) + \frac{\lambda}{|\Omega_Y|} \sum_{ij \in \Omega_Y} c_y(z_{ij}, z_{0ij})$$

$$\text{subject to} \quad \mathbf{Z} = \mathbf{Z}_0 + \mathbf{E} \tag{4}$$

$$\mathbf{Z}_1 = \mathbf{1}^\top.$$

Note that the constraint that $\mathbf{Z}_1$ remains equal to one is necessary for dealing with the bias $\mathbf{b}$ in (1). To avoid trivial solutions, large distortions of $\mathbf{Z}$ from known entries in $\mathbf{Z}_0$ are penalized according to losses $c_y(\cdot)$ and $c_x(\cdot)$: in [11], the former is defined as the Least Squares error, while the latter is a log loss to emphasize the error on entries switching classes as opposed to their absolute numerical difference. The parameters $\lambda, \mu$ are positive trade-off weights between better feature adaptation and label error correction. We note this problem is equivalent to

$$\underset{\mathbf{Z}}{\text{minimize}} \quad \mu \|\mathbf{Z}\|_* + \frac{1}{|\Omega_X|} \sum_{ij \in \Omega_X} c_x(z_{ij}, z_{0ij}) + \frac{\lambda}{|\Omega_Y|} \sum_{ij \in \Omega_Y} c_y(z_{ij}, z_{0ij})$$

$$\text{subject to} \quad \mathbf{Z}_1 = \mathbf{1}^\top \tag{5}$$

which can be solved using a Fixed Point Continuation method [18], described in Sec. 4.1.

## 4 Matrix completion for multi-label classification of visual data

In this section, we present the main contributions of this paper: the application of Matrix Completion to the multi-label image classification problem and its convergence proof. In the *bag of (visual) words* (BoW) model [24], visual data is encoded by the distribution of features among entries from a codebook. The codebook is typically created by clustering local feature representations such as SIFT [17] or GIST [21]. In this setting, the formulation MC-1 (5) is inadequate because it introduces negative values to the histograms in $\mathbf{Z}_X$. To address this issue, we replace the penalties used so they reflect the nature of data: we replace the Least-Squares penalty in $c_x(\cdot)$ by Pearson's $\chi^2$ distance, that takes into account the asymmetry in histogram data, as

$$\chi^2(\mathbf{z}_j, \mathbf{z}_{0j}) = \sum_{i=1}^{F} \chi_i^2(z_{ij}, z_{0ij}) = \sum_{i=1}^{F} \frac{(z_{ij} - z_{0ij})^2}{z_{ij} + z_{0ij}}. \tag{6}$$

As the modification to $c_x(\cdot)$ alone does not ensure that data retains its histogram nature, we add to (5) a constraint that all feature vectors in $\mathbf{Z}_X$ are either positive, resulting in the MC-Pos formulation

$$\begin{aligned} \underset{\mathbf{Z}}{\text{minimize}} \quad & \mu\|\mathbf{Z}\|_* + \frac{1}{|\Omega_X|} \sum_{ij \in \Omega_X} \chi_i^2(z_{ij}, z_{0ij}) + \frac{\lambda}{|\Omega_Y|} \sum_{ij \in \Omega_Y} c_y(z_{ij}, z_{0ij}) \\ \text{subject to} \quad & \mathbf{Z}_X \geq \mathbf{0} \\ & \mathbf{Z}_1 = \mathbf{1}^\top, \end{aligned} \tag{7}$$

or alternatively, that they belong to the Probability Simplex $\mathcal{P}$ (positive elements that sum to 1), resulting in the MC-Simplex formulation

$$\begin{aligned} \underset{\mathbf{Z}}{\text{minimize}} \quad & \mu\|\mathbf{Z}\|_* + \frac{1}{|\Omega_X|} \sum_{ij \in \Omega_X} \chi_i^2(z_{ij}, z_{0ij}) + \frac{\lambda}{|\Omega_Y|} \sum_{ij \in \Omega_Y} c_y(z_{ij}, z_{0ij}) \\ \text{subject to} \quad & \mathbf{Z}_X \in \mathcal{P} \\ & \mathbf{Z}_1 = \mathbf{1}^\top, \end{aligned} \tag{8}$$

depending on whether we wish to perform normalization on the data or not. Additionally, we note that the Log label error in $c_y(\cdot)$, albeit asymmetric, incurs in unnecessary penalization of entries belonging to the same class as the original entry (see Fig. 1). Therefore, we generalize this loss to progressively resemble smooth version of the Hinge loss, specified by the parameter $\gamma$ as

$$c_y(z_{ij}, z_{0ij}) = \frac{1}{\gamma} \log(1 + \exp(-\gamma z_{0ij} z_{ij})). \tag{9}$$

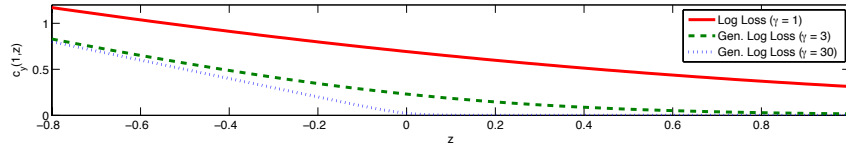

Figure 1: Comparison of Generalized Log loss with Log loss ($\gamma = 1$).

### 4.1 Fixed Point continuation (FPC) for MC-1

Albeit convex, the Nuclear Norm operator makes (5), (7), (8) not smooth. Since the natural reformulation of a Nuclear Norm minimization is a Semidefinite Program, existing off-the-shelf interior point methods are not applicable due to the large dimension of $\mathbf{Z}$. Thus, several methods have been devised to efficiently optimize this problem class [15, 6, 18, 25, 13, 1, 7, 2]. The FPC method [18], in particular, is comprised by a series of gradient updates $h(\cdot) = I(\cdot) - \tau g(\cdot)$ with step size $\tau$ and gradient $g(\cdot)$ given by the error penalizations $c_x(\cdot)$ and $c_y(\cdot)$. These steps are alternated with a shrinkage operator $S_\nu(\cdot) = \max(0, \cdot - \nu)$, applied to the singular values of the resulting matrix, so the rank is minimized. Provided $h(\cdot)$ is a contraction, this method provably converges to the optimal solution for the unconstrained problem. However, the formulation MC-1 (5) is constrained so in [11]

a projection step is added to the algorithm (see Alg. 1), whose convergence was only empirically verified. In this paper, we prove the convergence of FPC to the constrained problem class by using the fact that projections onto Convex sets are also non-expansive; thus, the composition of gradient, shrinkage and projection steps is also a contraction. Since the problem is convex, a unique fixed point exists in the optimal solution of the problem. First, let us write some preliminary results.

---

**Algorithm 1** FPC algorithm for solving MC-1 (5)

---

**Input:** Initial Matrix $\mathbf{Z}_0$
  Initialize $\mathbf{Z}$ as the rank-1 approximation of $\mathbf{Z}_0$
  **for** $\mu = \mu_1 > \mu_2 > \cdots > \mu_k$ **do**
    **while** Rel. Error $> \epsilon$ **do**
      Gradient Descent: $\mathbf{A} = h(\mathbf{A}) = \mathbf{Z} - \tau g(\mathbf{Z})$
      Shrink: $\mathbf{A} = \mathbf{U}\boldsymbol{\Sigma}\mathbf{V}^\top, \mathbf{Z} = \mathbf{U}S_{\tau\mu}(\boldsymbol{\Sigma})\mathbf{V}^\top$
      Project onto feasible set: $\mathbf{Z}_1 = \mathbf{1}^\top$
    **end while**
  **end for**
**Output:** Complete Matrix $\mathbf{Z}$

---

**Lemma 1** *Let $p_\mathcal{C}(\cdot)$ be a projection operator onto any given convex set $\mathcal{C}$. Then, $p_\mathcal{C}(\cdot)$ is non-expansive. Moreover, $\|p_\mathcal{C}(\mathbf{Z}) - p_\mathcal{C}(\mathbf{Z}^*)\| = \|\mathbf{Z} - \mathbf{Z}^*\|$ iff $p_\mathcal{C}(\mathbf{Z}) - p_\mathcal{C}(\mathbf{Z}^*) = \mathbf{Z} - \mathbf{Z}^*$.*

**Proof** For the first part, we apply the Cauchy-Schwarz inequality on the fact that (see [12, pg. 48])
$$\|p_\mathcal{C}(\mathbf{Z}) - p_\mathcal{C}(\mathbf{Z}^*)\|_F^2 \leq \langle p_\mathcal{C}(\mathbf{Z}) - p_\mathcal{C}(\mathbf{Z}^*), \mathbf{Z} - \mathbf{Z}^* \rangle. \tag{10}$$
For the second part, let us write
$$\|p_\mathcal{C}(\mathbf{Z}) - p_\mathcal{C}(\mathbf{Z}^*) - (\mathbf{Z} - \mathbf{Z}^*)\|_F^2 =$$
$$\|p_\mathcal{C}(\mathbf{Z}) - p_\mathcal{C}(\mathbf{Z}^*)\|_F^2 + \|\mathbf{Z} - \mathbf{Z}^*\|_F^2 - 2\langle p_\mathcal{C}(\mathbf{Z}) - p_\mathcal{C}(\mathbf{Z}^*), \mathbf{Z} - \mathbf{Z}^* \rangle, \tag{11}$$
where the inner product can be bounded by applying (10), yielding
$$\|p_\mathcal{C}(\mathbf{Z}) - p_\mathcal{C}(\mathbf{Z}^*) - (\mathbf{Z} - \mathbf{Z}^*)\|_F^2 \leq \|p_\mathcal{C}(\mathbf{Z}) - p_\mathcal{C}(\mathbf{Z}^*)\|_F^2 + \|\mathbf{Z} - \mathbf{Z}^*\|_F^2 - 2\|p_\mathcal{C}(\mathbf{Z}) - p_\mathcal{C}(\mathbf{Z}^*)\|_F^2. \tag{12}$$
Introducing our hypothesis $\|p_\mathcal{C}(\mathbf{Z}) - p_\mathcal{C}(\mathbf{Z}^*)\| = \|\mathbf{Z} - \mathbf{Z}^*\|$ into (12) yields
$$\|p_\mathcal{C}(\mathbf{Z}) - p_\mathcal{C}(\mathbf{Z}^*) - (\mathbf{Z} - \mathbf{Z}^*)\|_F^2 \leq 0, \tag{13}$$
from which we conclude an equality is in place.

**Theorem 2** *Let $\mathbf{Z}^*$ be an optimal solution to (5). Then $\mathbf{Z}$ is also an optimal solution if*
$$\|p_\mathcal{C}(S_\nu(h(\mathbf{Z}))) - p_\mathcal{C}(S_\nu(h(\mathbf{Z}^*)))\| = \|\mathbf{Z} - \mathbf{Z}^*\|. \tag{14}$$

**Proof** Using the non-expansiveness of operators $p_\mathcal{C}(\cdot)$, $S_\nu(\cdot)$ and $h(\cdot)$ (Lemma 1 and [18, Lemmas 1 and 2]), we can write
$$\|\mathbf{Z} - \mathbf{Z}^*\| = \|p_\mathcal{C}(S_\nu(h(\mathbf{Z}))) - p_\mathcal{C}(S_\nu(h(\mathbf{Z}^*)))\| \leq$$
$$\leq \|S_\nu(h(\mathbf{Z})) - S_\nu(h(\mathbf{Z}^*))\| \leq \quad \|h(\mathbf{Z}) - h(\mathbf{Z}^*))\| \leq \|\mathbf{Z} - \mathbf{Z}^*\|, \tag{15}$$
so we conclude the inequalities are equalities. Using the second part of the Lemmas, we get
$$p_\mathcal{C}(S_\nu(h(\mathbf{Z}^*))) - p_\mathcal{C}(S_\nu(h(\mathbf{Z}))) = S_\nu(h(\mathbf{Z}^*)) - S_\nu(h(\mathbf{Z})) = h(\mathbf{Z}^*) - h(\mathbf{Z}) = \mathbf{Z} - \mathbf{Z}^*. \tag{16}$$
Since $\mathbf{Z}^*$ is optimal, by the projected subgradient method, we have
$$p_\mathcal{C}(S_\nu(h(\mathbf{Z}^*))) = \mathbf{Z}^*, \tag{17}$$
which, in turn, implies that
$$p_\mathcal{C}(S_\nu(h(\mathbf{Z}))) = \mathbf{Z}, \tag{18}$$
from which we conclude $\mathbf{Z}$ is an optimal solution to (5).

We are now ready to prove the convergence of MC-1 to a fixed point $\mathbf{Z}^* = p_\mathcal{C}(S_\nu(h(\mathbf{Z}^*)))$, which allows us to state its result as an optimal solution of (5).

**Theorem 3** *The sequence $\{\mathbf{Z}^k\}$ generated by Alg. 1 converges to $\mathbf{Z}^*$, an optimal solution of (5).*

**Proof** Once we note the non-expansiveness of $p_\mathcal{C}(\cdot)$, $S_\nu(\cdot)$ and $h(\cdot)$ ensures the composite operator $p_\mathcal{C}(S_\nu(h(\cdot)))$ is also non-expansive, we can use the same rationale as in [18, Theorem 4].

## 4.2 Fixed Point Continuation for MC-Pos and MC-Simplex

The condition that $h(\cdot)$ is a contraction [18, Lemma 2] used for proving the convergence of Alg. 1 is still valid for the new loss functions proposed in (6) and (9), since the new gradient

$$g(z_{ij}) = \begin{cases} \frac{\lambda}{|\Omega_Y|} \frac{-z_{0ij}}{1+\exp{(\gamma z_{0ij} z_{ij})}} & \text{if } z_{ij} \in \Omega_Y, \\ \frac{1}{|\Omega_X|} \frac{z_{ij}^2 + 2z_{ij}z_{0ij} - 3z_{0ij}^2}{(z_{ij}+z_{0ij})^2} & \text{if } z_{ij} \in \Omega_X, \\ 0 & \text{otherwise} \end{cases} \tag{19}$$

is contractive, provided we choose a step size of $\tau \in [0, \min{(\frac{4|\Omega_Y|}{\lambda\gamma}, \tau_X|\Omega_X|)}]$. These values are easily obtained by noting the gradient of the Log loss function is Lipschitz continuous with $L = 0.25$ and choosing $\tau_X$ such that the $\chi^2$ error, for the Non-Negative Orthant, is Lipschitz continuous with $L = 1$. Key to the feasibility of (7) and (8) within this algorithmic framework, however, is an efficient way to project $\mathbf{Z}$ onto the newly defined constraint sets. While for MC-Pos (7) projecting a vector onto the Non-Negative Orthant is done in closed form by truncating negative components to zero, efficiently performing the projection onto the Probability Simplex in MC-Simplex (8) is not straightforward. We note, however, this is a projection onto a convex subset of an $\ell_1$ ball [9]. Therefore, we can explore the dual of the projection problem and use a sorting procedure to implement this projection in closed form, as described in Alg. 2. The final algorithms are summarized in Alg. 3 and Alg. 4.

---

**Algorithm 2** Projection of a vector onto probability Simplex

---

**Input:** Vector $\mathbf{v} \in \mathbb{R}^F$ to be projected
  Sort $\mathbf{v}$ into $\mu : \mu_1 \geq \mu_2 \geq ... \geq \mu_F$
  Find $\rho = \max\left\{ j \in n : \mu_j - \frac{1}{j}\left(\sum_{i=1}^{\rho} \mu_i - 1\right) > 0 \right\}$
  Compute $\theta = \frac{1}{\rho}\left(\sum_{i=1}^{\rho} \mu_i - 1\right)$
**Output:** $\mathbf{w}$ s.t. $w_i = \max\{v_i - \theta, 0\}$

---

---

**Algorithm 3** FPC Solver for MC-Pos (7)

---

**Input:** Initial Matrix $\mathbf{Z}_0$
  Initialize $\mathbf{Z}$ as the rank-1 approximation of $\mathbf{Z}_0$
  **for** $\mu = \mu_1 > \mu_2 > \cdots > \mu_k$ **do**
    **while** Rel. Error $> \epsilon$ **do**
      Gradient Descent: $\mathbf{A} = \mathbf{Z} - \tau g(\mathbf{Z})$
      Shrink 1: $\mathbf{A} = \mathbf{U}\boldsymbol{\Sigma}\mathbf{V}^\top$
      Shrink 2: $\mathbf{Z} = \mathbf{U}S_{\tau\mu}(\boldsymbol{\Sigma})\mathbf{V}^\top$
      Project $\mathbf{Z}_X$: $\mathbf{Z}_X = \max(\mathbf{Z}_X, \mathbf{0})$
      Project $\mathbf{Z}_1$: $\mathbf{Z}_1 = \mathbf{1}^\top$
    **end while**
  **end for**
**Output:** Complete Matrix $\mathbf{Z}$

---

**Algorithm 4** FPC Solver for MC-Simplex (8)

---

**Input:** Initial Matrix $\mathbf{Z}_0$
  Initialize $\mathbf{Z}$ as the rank-1 approximation of $\mathbf{Z}_0$
  **for** $\mu = \mu_1 > \mu_2 > \cdots > \mu_k$ **do**
    **while** Rel. Error $> \epsilon$ **do**
      Gradient Descent: $\mathbf{A} = \mathbf{Z} - \tau g(\mathbf{Z})$
      Shrink: $\mathbf{A} = \mathbf{U}\boldsymbol{\Sigma}\mathbf{V}^\top$
      Shrink: $\mathbf{Z} = \mathbf{U}S_{\tau\mu}(\boldsymbol{\Sigma})\mathbf{V}^\top$
      Project $\mathbf{Z}_X$ onto $\mathcal{P}$ (Alg. 2)
      Project $\mathbf{Z}_1$: $\mathbf{Z}_1 = \mathbf{1}^\top$
    **end while**
  **end for**
**Output:** Complete Matrix $\mathbf{Z}$

---

## 5 Experiments

This section presents the performance evaluation of the proposed algorithms MC-Pos (7) and MC-Simplex (8) in image categorization tasks. We compare our results with MC-1 (5) and standard discriminative and MIL approaches [30, 20, 27, 26, 33, 32] on three datasets: CMU-Face , MSRC and 15 Scene. For our algorithms and MC-1, the values considered for the parameter tuning were $\gamma \in \{1, 3, 30\}, \lambda \in [10^{-4}, 10^2]$. The continuation steps require a decreasing sequence of $\mu$, which we chose as $\mu_k = 0.25\mu_{k-1}$, stopping when $\mu = 10^{-12}$. We use $\mu_0 = 0.25\sigma_1$, where $\sigma_1$ is the largest singular value of $\mathbf{Z}_0$. Convergence was defined as a relative change in the objective function smaller than $10^{-2}$.

**CMU-Face dataset**  This dataset consists in 624 images of 20 subject faces with several expressions and poses, under two conditions: wearing sunglasses and not. We test single class classifica-

tion and localization. As in [20], our training set is built using images of the first 8 subjects (126 images with glasses and 128 without), leaving the remainder for testing (370, equally split among the classes). We describe each image by extracting 10000 SIFT features [17] at random scales and positions and quantizing them onto a 1000 visual codebook, obtained by performing hierarchical k-means clustering on 100000 features randomly selected from the training set. For this dataset, note that subjects were captured in a very similar environment, so the most discriminative part is the eye region. Thus, Nguyen *et al.* [20] argue that better results are obtained when the classifier training is restricted to that region. Since the face position varies, they propose using a Multiple Instance Learning framework (MIL-SegSVM), that localizes the most discriminative region in each image while learning a classifier to split both classes.

We compare the results of our classifier to the ones obtained by MIL-SegSVM as well as a Support Vector Machine. For the SVM, we either trained with the entire image information (SVM-Img) or with only the features extracted from the relevant, manually labeled, region of the eyes. For MC-1, MC-Pos and MC-Simplex, we proceed as follows. We fill $\mathbf{Z}$ with the label vector and the BoW histograms of each entire image and leave the test set labels $\mathbf{Y}_{tst}$ as unknown entries. For the MC-Simplex case, we preprocess $\mathbf{Z}$ by $\ell_1$-normalizing each histogram in $\mathbf{Z}_X$. This is done to avoid the Simplex projection picking a single bin and zeroing out the others, due to scale disparities in the bin counts. The obtained results are presented in Table 1, in terms of area under ROC curve (AUROC). These indicate both the fully supervised and the MIL approaches are more robust to the variability introduced by background noise, when compared to what is obtained when training without local-ization information (SVM-Img). However, this is done at either the cost of cumbersome labeling efforts or iteratively approximating the solution of MIL, an integer quadratic problem. By using Ma-trix Completion, in turn, we are able to surpass these classification scores by solving a single convex minimization, since our error term $\mathbf{E}$ removes noise introduced by non-discriminative parts of the image. To validate this hypothesis, we run a sliding window search in the images using the same size criteria of [20]. We search for the box having the normalized histogram most closely resemblant to the corrected version in $\mathbf{Z}_X$ according to the $\chi^2$ distance, and get the results shown in Fig. 2 (sim-ilar results were obtained using MC-Simplex). These show how the corrected histograms capture the semantic concept being trained. When comparing Matrix Completion approaches, we note that while the previous method MC-1 achieves competitive performance against previous baselines, it is outperformed by both MC-Pos, showing the improvement introduced by the domain knowledge constraints. Moreover, MC-1 does not allow to pursue further localization of the class representative since it introduces erroneous negative numbers in the histograms (Fig. 5).

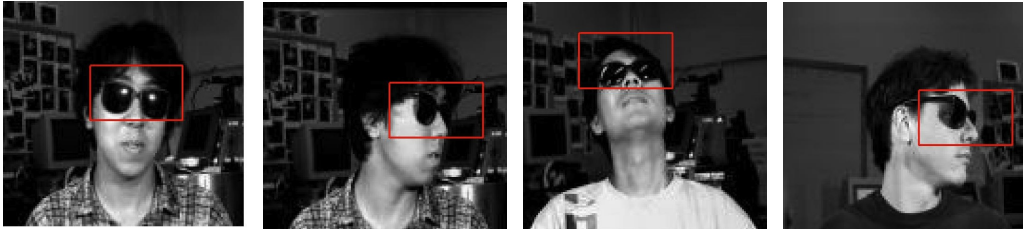

Figure 2: Histograms corrected by MC-Pos (7) preserve semantic meaning.

Table 1: AUROC result comparison for the CMU Face dataset.

| Method | AUROC |
|---|---|
| SVM-Img [20] | 0.90 |
| SVM-FS [20] | 0.94 |
| MIL-SegSVM [20] | 0.96 |
| MC-1 [11] | 0.96 |
| MC-Pos | **0.97** |
| MC-Simplex | 0.96 |

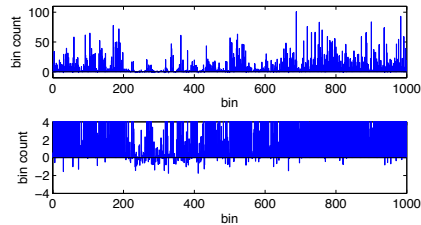

Figure 3: Erroneous histogram correction per-formed by MC-1 (5). Top: Global view. Bot-tom: Rescaling shows negative entries.

**MSRC dataset** Next, we run our method on a multi-label object recognition setting. The MSRC dataset consists of 591 real world images distributed among 21 classes, with an average of 3 classes present per image. We mimic the setup of [27] and use as features histograms of Textons [23] concatenated with histograms of colors in the L+U+V space. Our algorithm is given the task of classifying the presence of each object class in the images. We proceed as in the CMU-Face dataset.

In this dataset, we compare our formulations to MC-1 and several state-of-the-art approaches for categorization using Multiple-Label Multiple Instance Learning: Multiple Set Kernel MIL SVM (MSK-MIL) by Vijayanarasimhan *et al.* [27], Multi-label Multiple Instance Learning (ML-MIL) approach by Zha [32] and the Multi-task Random Texton Forest (MTL-RF) method of Vezhnevets *et al.* [26]. For localization, [32, 27] enumerate possible instances as the result of pre-segmenting images into a fixed number of parts, whereas [26] provides pixel level classification. The obtained average AUROC scores using 5-fold cross validation are shown in Table 2. Results show our methods significantly outperform MC-1. Moreover, MC-Simplex (8) outperforms results given by MIL techniques. Again, the fact that feature errors are corrected allows us to achieve good results while training with the entire image. This is opposed to relying on full blown pixel classification or segmentation techniques, which is still considered an open problem in Computer Vision. Moreover, we point out that MSK-MIL is a kernel approach as opposed to ours which, despite non-linear error penalizations, assumes a linear classification model in the feature space.

**15 Scene dataset** Finally, we test the performance of our algorithm for scene classification. Scenes differ from objects in the sense that they do not necessarily have a constrained physical location in the image. The 15 scene dataset is a multi-label dataset with 4485 images. According to the feature study in [30], we use GIST [21], the non-histogram feature achieving best results on this dataset. Notice that while not a BoW model, this feature represents the output energy of a bank of 24 filters, thus also positive. We run our algorithm on 10 folds, each comprised by 1500 training and 2985 test examples. The results on Table 3 show again that our method is able to achieve results comparable to state-of-the-art. One should note here that the state-of-the-art results are obtained by using a kernel space, whereas our method is essentially a linear technique aided by non linear error corrections. When we compare our results to using a linear kernel, MC-Simplex is able to achieve better performance. Relating to the results obtained for CMU-Face and MSRC datasets, we note that the roles of the MC-Pos and MC-Simplex are inverted, thus emphasizing the need for existence of models with and without normalization.

Table 2: 5-fold CV AUROC comparison for the MSRC dataset (Std. Dev. negligible at this precision).

| Method | Avg. AUROC |
|---|---|
| MSK-MIL[27] | 0.90 |
| ML-MIL [32] | 0.90 |
| MTL-RF [26] | 0.89 |
| MC-1 [11] | 0.87 |
| MC-Pos | **0.92** |
| MC-Simplex | 0.90 |

Table 3: 10-fold CV AUROC comparison for the 15 Scene dataset (Std. Dev. negligible at this precision).

| Method | Avg. AUROC |
|---|---|
| 1-vs-all Linear SVM [30] | 0.94 |
| 1-vs-all $\chi^2$ SVM [30] | **0.97** |
| MC-1 [11] | 0.90 |
| MC-Pos | 0.91 |
| MC-Simplex | 0.94 |

## 6 Conclusions

We presented two new convex methods for performing semi-supervised multi-label classification of histogram data, with proven convergence properties. Casting the classification under a Matrix Completion framework allows for easily handling of partial data and labels and robustness to outliers. Moreover, since histograms of full images contain the information for parts contained therein, the error embedded in our formulation is able to capture intra class variability arising from different backgrounds. Experiments show that our methods perform comparably to state-of-the-art MIL methods in several image datasets, surpassing them in several cases, without the need for precise localization of objects in the training set.

**Acknowledgements:** Support for this research was provided by the Portuguese Foundation for Science and Technology through the Carnegie Mellon Portugal program under the project FCT/CMU/P11. Partially funded by FCT project Printart PTDC/EEA-CRO/098822/2008. Fernando De la Torre is partially supported by Grant CPS-0931999 and NSF IIS-1116583. Any opinions, findings and conclusions or recommendations expressed in this material are those of the author(s) and do not necessarily reflect the views of the National Science Foundation.

## Footnotes

[1] Bold capital letters denote matrices (*e.g.*, $\mathbf{D}$), bold lower-case letters represent column vectors (*e.g.*, $\mathbf{d}$). All non-bold letters denote scalar variables. $\mathbf{d}_j$ is the $j^{th}$ column of the matrix $\mathbf{D}$. $d_{ij}$ denotes the scalar in the row $i$ and column $j$ of $\mathbf{D}$. $\langle \mathbf{d}_1, \mathbf{d}_2 \rangle$ denotes the inner product between two vectors $\mathbf{d}_1$ and $\mathbf{d}_2$. $\|\mathbf{d}\|_2^2 = \langle \mathbf{d}, \mathbf{d} \rangle = \sum_i d_i^2$ denotes the squared Euclidean Norm of the vector $\mathbf{d}$. $\text{tr}(\mathbf{A}) = \sum_i a_{ii}$ is the trace of the matrix $\mathbf{A}$. $\|\mathbf{A}\|_*$ designates the Nuclear Norm (sum of singular values) of $\mathbf{A}$. $\|\mathbf{A}\|_F^2 = \text{tr}(\mathbf{A}^\top \mathbf{A}) = \text{tr}(\mathbf{A}\mathbf{A}^\top)$ designates the squared Frobenius Norm of $\mathbf{A}$. $\mathbf{1}_k \in \mathbb{R}^{k \times 1}$ is a vector of ones, $\mathbf{0}_{k \times n} \in \mathbb{R}^{k \times n}$ is a matrix of zeros and $\mathbf{I}_k \in \mathbb{R}^{k \times k}$ denotes the identity matrix (dimensions are omitted when trivially inferred).

# References

[1] P. Aguiar, J. Xavier, and M. Stosic. Spectrally optimal factorization of incomplete matrices. In *CVPR*, 2008.

[2] L. Balzano, R. Nowak, and B. Recht. Online identification and tracking of subspaces from highly incomplete information. In *Proceedings of the 48th Annual Allerton Conference*, 2010.

[3] K. Barnard and D. Forsyth. Learning the semantics of words and pictures. In *ICCV*, 2001.

[4] T. L. Berg, A. C. Berg, J. Edwards, and D. A. Forsyth. Who's in the Picture? In *NIPS*, 2004.

[5] R. S. Cabral, J. P. Costeira, F. De la Torre, and A. Bernardino. Fast incremental method for matrix completion: an application to trajectory correction. In *ICIP*, 2011.

[6] J.-F. Cai, E. J. Candes, and Z. Shen. A singular value thresholding algorithm for matrix completion. *SIAM J. on Optimization*, 20(4):1956–1982, 2008.

[7] E. Candes and B. Recht. Exact low-rank matrix completion via convex optimization. In *Allerton*, 2008.

[8] Y. Dai, H. Li, and M. He. Element-wise factorization for n-view projective reconstruction. In *ECCV*, 2010.

[9] J. Duchi, S. Shalev-Shwartz, Y. Singer, and T. Chandra. Efficient projections onto the l1-ball for learning in high dimensions. In *ICML*, 2008.

[10] M. Fazel, H. Hindi, and S. P. Boyd. A rank minimization heuristic with application to minimum order system approximation. In *Proceedings American Control Conference*, 2001.

[11] A. B. Goldberg, X. Zhu, B. Recht, J. ming Xu, and R. Nowak. Transduction with matrix completion: Three birds with one stone. In *NIPS*, 2010.

[12] J.-B. Hiriart-Urruty and C. Lemaréchal. *Fundamentals of Convex Analysis*. Grundlehren der mathematien Wissenschaften. Springer-Verlag, New York–Heildelberg–Berlin, 2001.

[13] R. H. Keshavan, A. Montanari, and S. Oh. Matrix completion from a few entries. *IEEE Trans. Inf. Theor.*, 56:2980–2998, June 2010.

[14] V. Lavrenko, R. Manmatha, and J. Jeon. A model for learning the semantics of pictures. In *NIPS*, 2003.

[15] Z. Lin and M. Chen. The Augmented Lagrange Multiplier Method for Exact Recovery of Corrupted Low-Rank Matrices. *preprint*.

[16] G. Liu, Z. Lin, and Y. Yu. Robust subspace segmentation by low-rank representation. In *ICML*, 2010.

[17] D. G. Lowe. Distinctive image features from scale-invariant keypoints. *IJCV*, 60(2):91–110, 2004.

[18] S. Ma, D. Goldfarb, and L. Chen. Fixed point and bregman iterative methods for matrix rank minimization. *Mathematical Programming, to appear*.

[19] O. Maron and A. Ratan. Multiple-instance learning for natural scene classification. In *ICML*, 1998.

[20] M. H. Nguyen, L. Torresani, F. De la Torre, and C. Rother. Weakly supervised discriminative localization and classification: a joint learning process. In *ICCV*, 2009.

[21] A. Oliva and A. Torralba. Modeling the shape of the scene: A holistic representation of the spatial envelope. *IJCV*, 42:145–175, 2001.

[22] Y. Peng, A. Ganesh, J. Wright, W. Xu, and Y. Ma. Rasl: Robust alignment by sparse and low-rank decomposition for linearly correlated images. In *CVPR*, 2010.

[23] J. Shotton, J. M. Winn, C. Rother, and A. Criminisi. Textonboost: Joint appearance, shape and context modeling for multi-class object recognition and segmentation. In *ECCV*, 2006.

[24] J. Sivic and A. Zisserman. Video Google: A text retrieval approach to object matching in videos. In *CVPR*, 2003.

[25] K.-C. Toh and S. Yun. An accelerated proximal gradient algorithm for nuclear norm regularized least squares problems. *preprint*, 2009.

[26] A. Vezhnevets and J. Buhmann. Towards weakly supervised semantic segmentation by means of multiple instance and multitask learning. In *CVPR*, 2010.

[27] S. Vijayanarasimhan and K. Grauman. What's it going to cost you?: Predicting effort vs. informativeness for multi-label image annotations. In *CVPR*, 2009.

[28] H. Wang, C. Ding, and H. Huang. Multi-label linear discriminant analysis. In *ECCV*, 2010.

[29] J. Wright, A. Ganesh, S. Rao, and Y. Ma. Robust principal component analysis: Exact recovery of corrupted low-rank matrices by convex optimization. In *NIPS*, 2009.

[30] J. Xiao, J. Hays, K. A. Ehinger, A. Oliva, and A. Torralba. SUN database: Large-scale scene recognition from abbey to zoo. In *CVPR*, 2010.

[31] C. Yang, M. Dong, and J. Hua. Region-based image annotation using asymmetrical support vector machine-based multiple-instance learning. In *CVPR*, 2006.

[32] Z.-j. Zha, X.-s. Hua, T. Mei, J. Wang, and G.-j. Q. Zengfu. Joint multi-label multi-instance learning for image classification. In *CVPR*, 2008.

[33] Z.-h. Zhou and M. Zhang. Multi-instance multi-label learning with application to scene classification. In *NIPS*, 2006.

[34] G. Zhu, S. Yan, and Y. Ma. Image tag refinement towards low-rank, content-tag prior and error sparsity. In *ICMM*, 2010.

